# Boosting on manifolds: adaptive regularization of base classifiers

**Balázs Kégl and Ligen Wang**
Department of Computer Science and Operations Research, University of Montreal
CP 6128 succ. Centre-Ville, Montréal, Canada H3C 3J7
{kegl|wanglige}@iro.umontreal.ca

## Abstract

In this paper we propose to combine two powerful ideas, boosting and manifold learning. On the one hand, we improve ADABOOST by incorporating knowledge on the structure of the data into base classifier design and selection. On the other hand, we use ADABOOST's efficient learning mechanism to significantly improve supervised and semi-supervised algorithms proposed in the context of manifold learning. Beside the specific manifold-based penalization, the resulting algorithm also accommodates the boosting of a large family of regularized learning algorithms.

## 1 Introduction

ADABOOST [1] is one of the machine learning algorithms that have revolutionized pattern recognition technology in the last decade. The algorithm constructs a weighted linear combination of simple base classifiers in an iterative fashion. One of the remarkable properties of ADABOOST is that it is relatively immune to overfitting even after the training error has been driven to zero. However, it is now a common knowledge that ADABOOST *can* overfit if it is run long enough. The phenomenon is particularly pronounced on noisy data, so most of the effort to regularize ADABOOST has been devoted to make it tolerant to outliers by either "softening" the exponential cost function (e.g., [2]) or by explicitly detecting outliers and limiting their influence on the final classifier [3].

In this paper we propose a different approach based on complexity regularization. Rather than focusing on possibly noisy data points, we attempt to achieve regularization by favoring base classifiers that are smooth in a certain sense. The situation that motivated the algorithm is not when the data is noisy, rather when it has a certain structure that is ignored by ordinary ADABOOST. Consider, for example, the case when the data set is embedded in a high-dimensional space but concentrated around a low dimensional manifold (Figure 1(a)). ADABOOST will compare base classifiers based on solely their weighted errors so, implicitly, it will consider every base classifier having the same (usually low) complexity. On the other hand, intuitively, we may hope to achieve better generalization if we prefer base classifiers that "cut through" sparse regions to base classifiers that cut into "natural" clusters or cut the manifold several times. To formalize this intuition, we use the graph Laplacian regularizer proposed in connection to manifold learning [4] and spectral clustering [5] (Section 3). For binary base classifiers, this penalty is proportional to the number of edges of the neighborhood graph that the classifier cuts (Figure 1(b)).

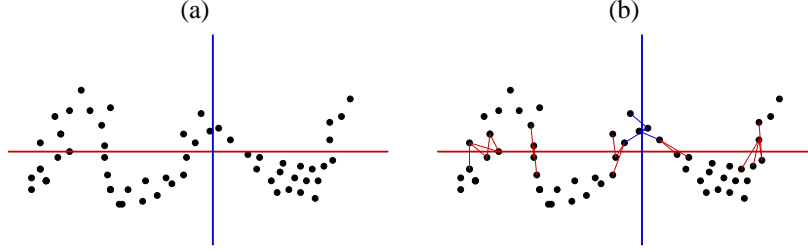

Figure 1: (a) Given the data, the vertical stump has a lower "effective" complexity than the horizontal stump. (b) The graph Laplacian penalty is proportional to the number of separated neighbors.

To incorporate this adaptive penalization of base classifiers into ADABOOST, we will turn to the *marginal* ADABOOST algorithm [6] also known as *arc-gv* [7]. This algorithm can be interpreted as ADABOOST with an $L_1$ weight decay on the base classifier coefficients with a weight decay coefficient $\theta$. The algorithm has been used to maximize the hard margin on the data [7, 6] and also for regularization [3]. The coefficient $\theta$ is adaptive in all these applications: in [7] and [6] it depends on the hard margin and the weighted error, respectively, whereas in [3] it is different for every training point and it quantifies the "noisiness" of the points. The idea of this paper is to make $\theta$ dependent on the individual base classifiers, in particular, to set $\theta$ to the regularization penalty of the base classifier. First, with this choice, the objective of base learning becomes standard regularized error minimization so the proposed algorithm accommodates the boosting of a large family of regularized learning algorithms. Second, the coefficients of the base classifiers are lowered proportionally with their complexity, which can be interpreted as an adaptive weight decay. The formulation can be also justified by theoretical arguments which are sketched after the formal description of the algorithm in Section 2.

Experimental results (Section 4) show that the regularized algorithm can improve generalization. Even when the improvement is not significant, the difference between the training error and the test error decreases significantly and the final classifier is much sparser than ADABOOST's solution, both of which indicate reduced overfitting. Since the Laplacian penalty can be computed without knowing the labels, the algorithm can also be used for semi-supervised learning. Experiments in this context show that algorithm besignificantly the semi-supervised algorithm proposed in [4].

## 2 The REGBOOST algorithm

For the formal description, let the training data be $D_n = \big((\mathbf{x}_1, y_1), \ldots, (\mathbf{x}_n, y_n)\big)$ where data points $(\mathbf{x}_i, y_i)$ are from the set $\mathbb{R}^d \times \{-1, 1\}$. The algorithm maintains a weight distribution $\mathbf{w}^{(t)} = \big(w_1^{(t)}, \ldots, w_n^{(t)}\big)$ over the data points. The weights are initialized uniformly in line 1 (Figure 2), and are updated in each iteration in line 10. We suppose that we are given a *base learner* algorithm $\text{BASE}\big(D_n, \mathbf{w}, P(\cdot)\big)$ that, in each iteration $t$, returns a *base classifier* $h^{(t)}$ coming from a subset of $\mathcal{H} = \big\{h : \mathbb{R}^d \mapsto \{-1, 1\}\big\}$. In ADABOOST, the goal of the base classifier is to minimize the *weighted error*

$$\epsilon = \epsilon^{(t)}(h) = \sum_{i=1}^{n} w_i^{(t)} I\{h(\mathbf{x}_i) \neq y_i\},\ ^{12}$$

$$\text{REGBOOST}\Big(D_n, \text{BASE}(\cdot, \cdot, \cdot), P(\cdot), \lambda, T\Big)$$

1      $\mathbf{w} \leftarrow (1/n, \ldots, 1/n)$

2      **for** $t \leftarrow 1$ **to** $T$

3            $h^{(t)} \leftarrow \text{BASE}\big(D_n, \mathbf{w}^{(t)}, P(\cdot)\big)$

4            $\gamma^{(t)} \leftarrow \sum_{i=1}^{n} w_i^{(t)} h^{(t)}(\mathbf{x}_i) y_i$        ▷ *edge*

5            $\theta^{(t)} \leftarrow 2\lambda P(h^{(t)})$       ▷ *edge offset*

6            $\alpha^{(t)} \leftarrow \frac{1}{2} \ln\left(\frac{1 + \gamma^{(t)}}{1 - \gamma^{(t)}} \cdot \frac{1 - \theta^{(t)}}{1 + \theta^{(t)}}\right)$     ▷ *base coefficient*

7            **if** $\alpha^{(t)} \leq 0$      ▷ $\Longleftrightarrow$ *base error* $\geq (1 - \theta^{(t)})/2$

8                 **return** $f^{(t-1)}(\cdot) = \sum_{j=1}^{t-1} \alpha^{(j)} h^{(j)}(\cdot)$

9            **for** $i \leftarrow 1$ **to** $n$

10                 $w_i^{(t+1)} \leftarrow w_i^{(t)} \dfrac{\exp\big(-\alpha^{(t)} h^{(t)}(\mathbf{x}_i) y_i\big)}{\sum_{j=1}^{n} w_j^{(t)} \exp\big(-\alpha^{(t)} h^{(t)}(\mathbf{x}_j) y_j\big)}$

11     **return** $f^{(T)}(\cdot) = \sum_{t=1}^{T} \alpha^{(t)} h^{(t)}(\cdot)$

Figure 2: The pseudocode of the REGBOOST algorithm with binary base classifiers. $D_n$ is the training data, BASE is the base learner, $P$ is the penalty functional, $\lambda$ is the penalty coefficient, and $T$ is the number of iterations.

which is equivalent to maximizing the *edge* $\gamma = 1 - 2\epsilon = \sum_{i=1}^{n} w_i^{(t)} h(\mathbf{x}_i) y_i$. The goal of REGBOOST's base learner is to minimize the *penalized cost*

$$R_1(h) = \epsilon(h) + \lambda P(h) = \frac{1}{2} - \frac{1}{2}(\gamma - \theta), \tag{1}$$

where $P : \mathcal{H} \mapsto \mathbb{R}$ is an arbitrary *penalty functional* or *regularization operator*, provided to REGBOOST and to the base learner, $\lambda$ is the *penalty coefficient*, and $\theta = 2\lambda P(h)$ is the *edge offset*. Intuitively, the edge $\gamma$ quantifies by how much $h$ *is* better than a random guess, while the edge offset $\theta$ indicates by how much $h^{(t)}$ *must be* better than a random guess. This means that for complex base classifiers (with large penalties), we require a better base classification than for simple classifiers. The main advantage of $R_1$ is that it has the form of conventional regularized error minimization, so it accommodates the boosting of all learning algorithms that minimize an error functional of this form (e.g., neural networks with weight decay). However, the minimization of $R_1$ is suboptimal from boosting's point of view.[3] If computationally possible, the base learner should minimize

$$R_2(h) = 2\sqrt{\left(\frac{1-\epsilon}{1+\theta}\right)^{1+\theta} \left(\frac{\epsilon}{1-\theta}\right)^{1-\theta}} = \sqrt{\left(\frac{1+\gamma}{1+\theta}\right)^{1+\theta} \left(\frac{1-\gamma}{1-\theta}\right)^{1-\theta}}. \tag{2}$$

After computing the edge and the edge offset in lines 4 and 5, the algorithm sets the coefficient $\alpha^{(t)}$ of the base classifier $h^{(t)}$ to

$$\alpha^{(t)} = \frac{1}{2} \ln \left( \frac{1 + \gamma^{(t)}}{1 - \gamma^{(t)}} \right) - \frac{1}{2} \ln \left( \frac{1 + \theta^{(t)}}{1 - \theta^{(t)}} \right). \tag{3}$$

In line 11, the algorithm returns the weighted average of the base classifiers $f^{(T)}(\cdot) = \sum_{t=1}^{T} \alpha^{(t)} h^{(t)}(\cdot)$ as the *combined classifier*, and uses the sign of $f^{(T)}(\mathbf{x})$ to classify $\mathbf{x}$. The algorithm must terminate if $\alpha^{(t)} \leq 0$ which is equivalent to $\gamma^{(t)} \leq \theta^{(t)}$ and to $\epsilon^{(t)} \geq (1-\theta^{(t)})/2$.[4] In this case, the algorithm returns the actual combined classifier in line 8. This means that either the capacity of the set of base classifiers is too small ($\gamma^{(t)}$ is small), or the penalty is too high ($\theta^{(t)}$ is high), so we cannot find a new base classifier that would improve the combined classifier. Note that the algorithm is formally equivalent to ADABOOST if $\theta^{(t)} \equiv 0$ and to marginal ADABOOST if $\theta^{(t)} \equiv \theta$ is constant.

For the analysis of the algorithm, we first define the *unnormalized margin* achieved by $f^{(T)}$ on $(\mathbf{x}_i, y_i)$ as

$$\rho_i = f^{(T)}(\mathbf{x}_i) y_i,$$

and the (normalized) *margin* as

$$\widetilde{\rho}_i = \frac{\rho_i}{\|\boldsymbol{\alpha}\|_1} = \frac{\sum_{t=1}^{T} \alpha^{(t)} h^{(t)}(\mathbf{x}_i) y_i}{\sum_{t=1}^{T} \alpha^{(t)}}, \tag{4}$$

where $\|\boldsymbol{\alpha}\|_1 = \sum_{t=1}^{T} \alpha^{(t)}$ is the $L_1$ norm of the coefficient vector. Let the *average penalty* or *margin offset* be defined as the average edge offset

$$\bar{\theta} = \frac{\sum_{t=1}^{T} \alpha^{(t)} \theta^{(t)}}{\sum_{t=1}^{T} \alpha^{(t)}}. \tag{5}$$

The following theorem upper bounds the *marginal training error*

$$L^{(\bar{\theta})}(f^{(T)}) = \frac{1}{n} \sum_{i=1}^{n} I\left\{ \widetilde{\rho}_i < \bar{\theta} \right\} \tag{6}$$

achieved by the combined classifier $f^{(T)}$ that REGBOOST outputs.

**Theorem 1** *Let* $\theta^{(t)} = 2\lambda P(h^{(t)})$, *let* $\bar{\theta}$ *and* $L^{(\bar{\theta})}(f^{(T)})$ *be as defined in (5) and (6), respectively. Let* $w_i^{(t)}$ *be the weight of training point* $(\mathbf{x}_i, y_i)$ *after the tth iteration (updated in line 10 in Figure 2), and let* $\alpha^{(t)}$ *be the weight of the base regressor* $h^{(t)}(\cdot)$ *(computed in line 6 in Figure 2). Then*

$$L^{(\bar{\theta})}(f^{(T)}) \leq \prod_{t=1}^{T} e^{\theta^{(t)} \alpha^{(t)}} \sum_{i=1}^{n} w_i^{(t)} e^{-\alpha^{(t)} h^{(t)}(\mathbf{x}_i) y_i} \triangleq \prod_{t=1}^{T} E^{(t)}\left(\alpha^{(t)}, h^{(t)}\right). \tag{7}$$

**Proof.** The proof is an extension of the proof of Theorem 5 in [8].

$$L^{(\bar{\theta})}(f^{(T)}) = \frac{1}{n} \sum_{i=1}^{n} I\left\{ \bar{\theta} \sum_{t=1}^{T} \alpha^{(t)} - \sum_{t=1}^{T} \alpha^{(t)} h^{(t)}(\mathbf{x}_i) y_i \geq 0 \right\} \tag{8}$$

$$\leq \frac{1}{n} \sum_{i=1}^{n} e^{\bar{\theta} \sum_{t=1}^{T} \alpha^{(t)} - \sum_{t=1}^{T} \alpha^{(t)} h^{(t)}(\mathbf{x}_i) y_i} \tag{9}$$

$$= e^{\bar{\theta} \sum_{t=1}^{T} \alpha^{(t)}} \prod_{t=1}^{T} \sum_{j=1}^{n} w_j^{(t)} e^{-\alpha^{(t)} h^{(t)}(\mathbf{x}_j) y_j} \sum_{i=1}^{n} w_i^{(T+1)}. \tag{10}$$

In (8) we used the definitions (6) and (4), the inequality (9) holds since $e^x \geq I\{x \geq 0\}$, and we obtained (10) by recursively applying line 10 in Figure 2. The theorem follows by the definition (5) and since $\sum_{i=1}^{n} w_i^{(T+1)} = 1$. $\qquad\square$

First note that Theorem 1 explains the base objectives (1) and (2) and the base coefficient (3). The goal of REGBOOST is the greedy minimization of the exponential bound in (7), that is, in each iteration we attempt to minimize $E^{(t)}(\alpha, h)$. Given $h^{(t)}$, $E^{(t)}(\alpha, h^{(t)})$ is minimized by (3), and with this choice for $\alpha^{(t)}$, $R_2(h) = E^{(t)}(\alpha^{(t)}, h)$, so the base learner should attempt to minimize $R_2(h)$. If this is computationally impossible, we follow Mason et al.'s functional gradient descent approach [2], that is, we find $h^{(t)}$ by maximizing the negative gradient $-\frac{\partial E^{(t)}(\alpha, h)}{\partial \alpha}$ in $\alpha = 0$. Since $-\frac{\partial E^{(t)}(\alpha, h)}{\partial \alpha}\Big|_{\alpha=0} = \gamma - \theta$, this criterion is equivalent to the minimization of $R_1(h)$.[5]

Theorem 1 also suggests various interpretations of REGBOOST which indicate why it would indeed achieve regularization. First, by (9) it can be seen that REGBOOST directly minimizes

$$\frac{1}{n} \sum_{i=1}^{n} \exp\left(-\rho_i + \bar{\theta}\|\boldsymbol{\alpha}\|_1\right),$$

which can be interpreted as an exponential cost on the unnormalized margin with an $L_1$ weight decay. The weight decay coefficient $\bar{\theta}$ is proportional to the average complexity of the base classifiers. Second, Theorem 1 also indicates that REGBOOST indirectly minimizes the marginal error $L^{(\bar{\theta})}(f^{(T)})$ (6) where the margin parameter $\bar{\theta}$, again, is moving adaptively with the average complexity of the base classifiers. This explanation is supported by theoretical results that bound the generalization error in terms of the marginal error (e.g., Theorem 2 in [8]). The third explanation is based on results that show that the difference between the marginal error and the generalization error can be upper bounded in terms of the complexity of the base classifier class $\mathcal{H}$ (e.g., Theorem 4 in [9]). By imposing a non-zero penalty on the base classifiers, we can reduce the pool of admissible functions to those of which the edge $\gamma$ is larger than the edge offset $\theta$. Although the theoretical results do not apply directly, they support the empirical evidence (Section 4) that indicate that the reduction of the pool of admissible base classifiers and the sparsity of the combined classifier play an important role in decreasing the generalization error.

Finally note that the algorithm can be easily extended to real-valued base classifiers along the lines of [10] and to regression by using the algorithm proposed in [11]. If base classifiers come from the set $\{h : \mathbb{R}^d \mapsto \mathbb{R}\}$, we can only use the base objective $R_1(h)$ (1), and the analytical solution (3) for the base coefficients $\alpha^{(t)}$ must be replaced by a simple numerical minimization (line search) of $E^{(t)}(\alpha, h^{(t)})$.[6] In the case of regression, the binary cost function $I\{h(\mathbf{x}) \neq y\}$ should be replaced by an appropriate regression cost (e.g., quadratic), and the final regressor should be the weighted median of the base regressors instead of their weighted average.

## 3 The graph Laplacian regularizer

The algorithm can be used with any regularized base learner that optimizes a penalized cost of the form (1). In this paper we apply a smoothness functional based on the graph

Laplacian operator, proposed in a similar context by [4]. The advantage of this penalty is that it is relatively simple to compute for enumerable base classifiers (e.g., decision stumps or decision trees) and that it suits applications where the data exhibits a low dimensional manifold structure.

Formally, let $G = (\mathcal{V}, \mathcal{E})$ be the *neighborhood graph* of the training set where the vertex set $\mathcal{V} = \{\mathbf{x}_1, \ldots, \mathbf{x}_n\}$ is identical to the set of observations, and the edge set $\mathcal{E}$ contains pairs of "neighboring" vertices $(\mathbf{x}_i, \mathbf{x}_j)$ such that either $\|\mathbf{x}_i - \mathbf{x}_j\| < r$ or $\mathbf{x}_i$ ($\mathbf{x}_j$) is among the $k$ nearest neighbors of $\mathbf{x}_j$ ($\mathbf{x}_i$) where $r$ or $k$ is fixed. This graph plays a crucial role in several recently developed dimensionality reduction methods since it approximates the natural topology of the data if it is confined to a low-dimensional smooth manifold in the embedding space. To penalize base classifiers that cut through dense regions, we use the smoothness functional

$$P_{\mathcal{L}}(h) = \frac{1}{2|\mathbf{W}|} \sum_{i=1}^{n} \sum_{j=i+1}^{n} \left( h(\mathbf{x}_i) - h(\mathbf{x}_j) \right)^2 W_{ij},$$

where $\mathbf{W}$ is the adjacency matrix of $G$, that is, $W_{ij} = I\{(\mathbf{x}_i, \mathbf{x}_j) \in \mathcal{E}\}$, and $2|\mathbf{W}| = 2\sum_{i=1}^{n} \sum_{j=1}^{n} W_{ij}$ is a normalizing factor so that $0 \leq P_{\mathcal{L}}(h) \leq 1$.[7] For binary base classifiers, $P_{\mathcal{L}}(h)$ is proportional to the number of separated neighbors, that is, the number of connected pairs that are classified differently by $h$. Let the diagonal matrix $\mathbf{D}$ defined by $D_{ii} = \sum_{j=1}^{n} W_{ij}$, and let $\mathbf{L} = \mathbf{D} - \mathbf{W}$ be the *graph Laplacian* of $G$. Then it is easy to see that

$$2|\mathbf{W}|P_{\mathcal{L}}(h) = \mathbf{h}\mathbf{L}\mathbf{h}^T = \langle \mathbf{h}, \mathbf{L}\mathbf{h} \rangle = \sum_{j=1}^{n} \lambda_i \langle \mathbf{h}, \mathbf{e}_i \rangle,$$

where $\mathbf{h} = \left( h(\mathbf{x}_1), \ldots, h(\mathbf{x}_n) \right)$, and $\mathbf{e}_i$ and $\lambda_i$ are the (normalized) eigenvectors and eigenvalues of $\mathbf{L}$, that is, $\mathbf{L}\mathbf{e}_i = \lambda_i \mathbf{e}_i$, $\|\mathbf{e}_i\| = 1$. Since $\mathbf{L}$ is positive definite, all the eigenvalues are non-negative. The eigenvectors with the smallest eigenvalues can be considered as the "smoothest" functions on the neighborhood graph. Based on this observation, [4] proposed to learn a linear combination of a small number of the eigenvectors with the smallest eigenvalues. One problem of this approach is that the out-of-sample extension of the obtained classifier is non-trivial since the base functions are only known at the data points that participated in forming the neighborhood graph, so it can only be used in a semi-supervised settings (when unlabeled test points are known before the learning). Our approach is based on the same intuition, but instead of looking for a linear combination of the eigenvectors, we form a linear combination of *known* base functions and penalize them according to their smoothness on the underlying manifold. So, beside semi-supervised learning (explored in Section 4), our algorithm can also be used to classify out-of-sample test observations.

The penalty functional can also be justified from the point of view of spectral clustering [5]. The eigenvectors of $\mathbf{L}$ with the smallest eigenvalues[8] represent "natural" clusters in the data set, so $P_{\mathcal{L}}(h)$ is small if $h$ is aligned with these eigenvectors, and $P_{\mathcal{L}}(h)$ is large if $h$ splits the corresponding clusters.

## 4 Experiments

In this section we present experimental results on four UCI benchmark datasets. The results are preliminary in the sense that we only validated the penalty coefficient $\lambda$, and did not optimize the number of neighbors (set to $k = 8$) and the weighting scheme of the edges of the neighborhood graph ($W_{ij} = 0$ or $1$). We used decision stumps as base classifiers, 10-fold cross validation for estimating errors, and 5-fold cross validation for determining $\lambda$. The results (Figure 3(a)-(d) and Table 1) show that the REGBOOST consistently improves generalization. Although the improvement is within the standard deviation, the difference between the test and the training error decreases significantly in two of the four experiments, which indicates reduced overfitting. The final classifier is also significantly sparser after 1000 iterations (last two columns of Table 1). To measure how the penalty affects the base classifier pool, in each iteration we calculated the number of admissible base classifiers relative to the total number of stumps considered by ADABOOST. Figure 3(e) shows that, as expected, REGBOOST traverses only a (sometimes quite small) subset of the base classifier space.

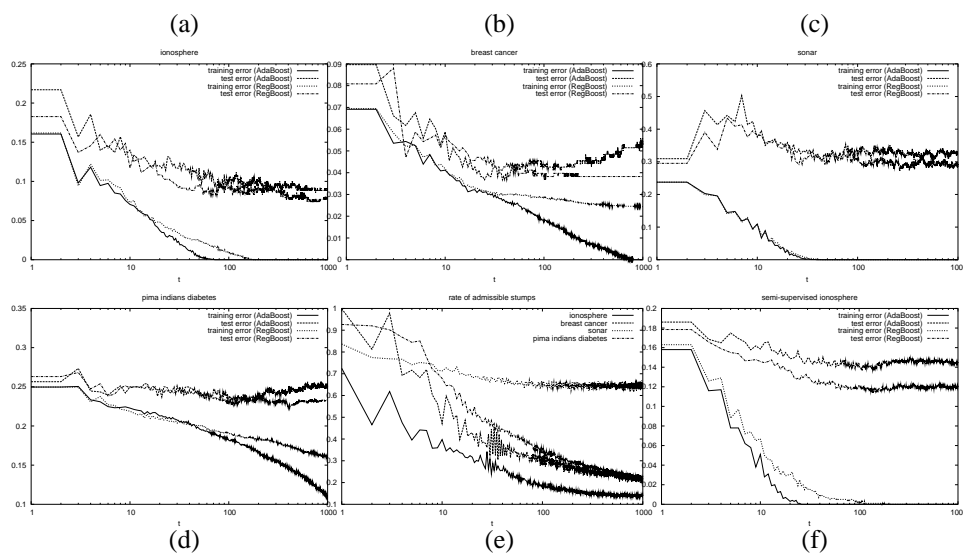

Figure 3: **Learning curves**. Test and training errors for the (a) ionosphere, (b) breast cancer, (c) sonar, and (d) Pima Indians diabetes data sets. (e) Rate of admissible stumps. (f) Test and training errors for the ionosphere data set with 100 labeled and 251 unlabeled data points.

| data set | training error | | test error | | # of stumps | |
|---|---|---|---|---|---|---|
| | ADAB | REGB | ADAB | REGB | ADAB | REGB |
| ionosphere | 0% | 0% | 9.14% (7.1) | 7.7% (6.0) | 182 | 114 |
| breast cancer | 0% | 2.44% | 5.29% (3.5) | 3.82% (3.7) | 58 | 30 |
| sonar | 0% | 0% | 32.5% (19.8) | 29.8% (18.8) | 234 | 199 |
| Pima Indians | 10.9% | 16.0% | 25.3% (5.3) | 23.3% (6.8) | 175 | 91 |

Table 1: Errors rates and number of base classifiers after 1000 iterations.

Since the Laplacian penalty can be computed without knowing the labels, the algorithm can also be used for semi-supervised learning. Figure 3(f) shows the results when only a subset of the training points are labeled. In this case, REGBOOST can use the combined data set to calculate the penalty, whereas both algorithms can use only the labeled points

to determine the base errors. Figure 3(f) indicates that REGBOOST has a clear advantage here. REGBOOST is also far better than the semi-supervised algorithm proposed in [12] (their best test error using the same settings is $18\%$).

## 5    Conclusion

In this paper we proposed to combine two powerful ideas, boosting and manifold learning. The algorithm can be used to boost any regularized base learner. Experimental results indicate that REGBOOST slightly improves ADABOOST by incorporating knowledge on the structure of the data into base classifier selection. REGBOOST also significantly improves a recently proposed semi-supervised algorithm based on the same regularizer. In the immediate future our goal is to conduct a larger scale experimental study in which we optimize all the parameters of the algorithm, and compare it not only to ADABOOST, but also to marginal ADABOOST, that is, REGBOOST with a constant penalty $\theta$. Marginal ADABOOST might exhibit a similar behavior on the supervised task (sparsity, reduced number of admissible base classifiers), however, it can not be used to semi-supervised learning. We also plan to experiment with other penalties which are computationally less costly than the Laplacian penalty.

## Footnotes

[1] The indicator function $I\{A\}$ is 1 if its argument $A$ is true and 0 otherwise.

[2] We will omit the iteration index $^{(t)}$ and the argument $^{(h)}$ where it does not cause confusion.

[3]This statement along with the formulae for $R_1$, $R_2$, and $\alpha^{(t)}$ are explained formally after Theorem 1.

[4]Strictly speaking, $\alpha^{(t)} = 0$ could be allowed but in this case the $\alpha^{(t)}$ would remain 0 forever so it makes no sense to continue.

[5]Note that if $\theta$ is constant (ADABOOST or marginal ADABOOST), the minimization of $R_1(h)$ and $R_2(h)$ leads to the same solution, namely, to the base classifier that minimizes the weighted error $\epsilon$. This is no more the case if $\theta$ depends on $h$.

[6]As a side remark, note that applying a non-zero (even constant) penalty $\theta$ would provide an alternative solution to the singularity problem ($\alpha^{(t)} = \infty$) in the abstaining base classifier model of [10].

[7]Another variant (that we did not explore in this paper) is to weight edges decreasingly with their lengths.

[8]Starting from the second smallest; the smallest is 0 and it corresponds to the constant function. Also note that spectral clustering usually uses the eigenvectors of the *normalized* Laplacian $\widetilde{L} = \mathbf{D}^{-1/2}\mathbf{L}\mathbf{D}^{-1/2}$. Nevertheless, if the neighborhood graph is constructed by connecting a fixed number of nearest neighbors, $D_{ii}$ is approximately constant, so the eigenvectors of $L$ and $\widetilde{L}$ are approximately equal.

## References

[1]  Y. Freund and R. E. Schapire, "A decision-theoretic generalization of on-line learning and an application to boosting," *Journal of Computer and System Sciences*, vol. 55, pp. 119–139, 1997.

[2]  L. Mason, P. Bartlett, J. Baxter, and M. Frean, "Boosting algorithms as gradient descent," in *Advances in Neural Information Processing Systems*. 2000, vol. 12, pp. 512–518, The MIT Press.

[3]  G. Rätsch, T. Onoda, and K.-R. Müller, "Soft margins for AdaBoost," *Machine Learning*, vol. 42, no. 3, pp. 287–320, 2001.

[4]  M. Belkin and P. Niyogi, "Semi-supervised learning on Riemannian manifolds," *Machine Learning, to appear*, 2004.

[5]  J. Shi and J. Malik, "Normalized cuts and image segmentation," *IEEE Transactions on Pattern Analysis and Machine Intelligence*, vol. 22, no. 8, pp. 888–905, 2000.

[6]  G. Rätsch and M. K. Warmuth, "Maximizing the margin with boosting," in *Proceedings of the 15th Conference on Computational Learning Theory*, 2002.

[7]  L. Breiman, "Prediction games and arcing classifiers," *Neural Computation*, vol. 11, pp. 1493–1518, 1999.

[8]  R. E. Schapire, Y. Freund, P. Bartlett, and W. S. Lee, "Boosting the margin: a new explanation for the effectiveness of voting methods," *Annals of Statistics*, vol. 26, no. 5, pp. 1651–1686, 1998.

[9]  A. Antos, B. Kégl, T. Linder, and G. Lugosi, "Data-dependent margin-based generalization bounds for classification," *Journal of Machine Learning Research*, pp. 73–98, 2002.

[10]  R. E. Schapire and Y. Singer, "Improved boosting algorithms using confidence-rated predictions," *Machine Learning*, vol. 37, no. 3, pp. 297–336, 1999.

[11]  B. Kégl, "Robust regression by boosting the median," in *Proceedings of the 16th Conference on Computational Learning Theory*, Washington, D.C., 2003, pp. 258–272.

[12]  M. Belkin, I. Matveeva, and P. Niyogi, "Regression and regularization on large graphs," in *Proceedings of the 17th Conference on Computational Learning Theory*, 2004.
